# TrueSkill Through Time:
# Revisiting the History of Chess

**Pierre Dangauthier**
INRIA Rhone Alpes
Grenoble, France
*pierre.dangauthier@imag.fr*

**Ralf Herbrich**
Microsoft Research Ltd.
Cambridge, UK
*rherb@microsoft.com*

**Tom Minka**
Microsoft Research Ltd.
Cambridge, UK
*minka@microsoft.com*

**Thore Graepel**
Microsoft Research Ltd.
Cambridge, UK
*thoreg@microsoft.com*

## Abstract

We extend the Bayesian skill rating system TrueSkill to infer entire time series of skills of players by smoothing through time instead of filtering. The skill of each participating player, say, every year is represented by a latent skill variable which is affected by the relevant game outcomes that year, and coupled with the skill variables of the previous and subsequent year. Inference in the resulting factor graph is carried out by approximate message passing (EP) along the time series of skills. As before the system tracks the uncertainty about player skills, explicitly models draws, can deal with any number of competing entities and can infer individual skills from team results. We extend the system to estimate player-specific draw margins. Based on these models we present an analysis of the skill curves of important players in the history of chess over the past 150 years. Results include plots of players' lifetime skill development as well as the ability to compare the skills of different players across time. Our results indicate that a) the overall playing strength has increased over the past 150 years, and b) that modelling a player's ability to force a draw provides significantly better predictive power.

## 1   Introduction

Competitive games and sports can benefit from statistical skill ratings for use in match-making as well as for providing criteria for the admission to tournaments. From a historical perspective, skill ratings also provide information about the general development of skill within the discipline or for a particular group of interest. Also, they can give a fascinating narrative about the key players in a given discipline, allowing a glimpse at their rise and fall or their struggle against their contemporaries.

In order to provide good estimates of the current skill level of players skill rating systems have traditionally been designed as filters that combine a new game outcome with knowledge about a player's skill from the past to obtain a new estimate. In contrast, when taking a historical view we would like to infer the skill of a player at a given point in the past when both their past as well as their future achievements are known.

The best known such skill filter based rating system is the *Elo system* [3] developed by Arpad Elo in 1959 and adopted by the World Chess Federation FIDE in 1970 [4]. Elo models the

probability of the game outcome as $P\left(1 \text{ wins over } 2|s_1, s_2\right) := \Phi\left(\frac{s_1 - s_2}{\sqrt{2}\beta}\right)$ where $s_1$ and $s_2$ are the skill ratings of each player, $\Phi$ denotes the cumulative density of a zero-mean unit-variance Gaussian and $\beta$ is the assumed variability of performance around skill. Denote the game outcomes by $y = +1$ if player 1 wins, $y = -1$ if player 2 wins and $y = 0$ if a draw occurs. Then the resulting (linearised) Elo update is given by $s_1 \leftarrow s_1 + y\Delta$, $s_2 \leftarrow s_2 - y\Delta$ and

$$\Delta = \underbrace{\alpha\beta\sqrt{\pi}}_{K-\text{Factor}}\left(\frac{y+1}{2} - P\left(1 \text{ wins over } 2|s_1, s_2\right)\right),$$

where $0 < \alpha < 1$ determines how much the filter weighs the new evidence versus the old estimate.

The TrueSkill rating system [6] improves on the Elo system in a number of ways. TrueSkill's current belief about a player's skill is represented by a Gaussian distribution with mean $\mu$ and variance $\sigma^2$. As a consequence, TrueSkill does not require a provisional rating period and converges to the true skills of players very quickly. Also, in contrast to Elo, TrueSkill explicitly models the probability of draws. Crucially for its application in the Xbox Live online gaming system (see [6] for details) it can also infer skills from games with more than two participating entities and infers individual players' skills from the outcomes of team games.

As a skill rating and matchmaking system TrueSkill operates as a filter as discussed above. However, due to its fully probabilistic formulation it is possible to extend Trueskill to perform smoothing on a time series of player skills. In this paper we extend TrueSkill to provide accurate estimates of the past skill levels of players at any point in time taking into account both their past and their future achievements. We carry out a large-scale analysis of about 3.5 million games of chess played over the last 150 years.

The paper is structured as follows. In Section 2 we review previous work on historical chess ratings. In Section 3 we present two models for historical ratings through time, one assuming a fixed draw margin and one estimating the draw margin per player per year. We indicate how large scale approximate message passing (EP) can be used to efficiently perform inference in these huge models. In Section 4 we present experimental results on a huge data set from ChessBase with over 3.5 million games and gain some fascinating chess specific insights from the data.

## 2 Previous Work on Historical Chess Ratings

Estimating players' skills in retrospective allows one to take into account more information and hence can be expected to lead to more precise estimates. The pioneer in this field was Arpad Elo himself, when he encountered the necessity of initializing the skill values of the Elo system when it was first deployed. To that end he fitted a smooth curve to skill estimates from five-year periods; however little is known about the details of his method [3].

Probably best known in the chess community is the Chessmetrics system [8], which aims at improving the Elo scores by attempting to obtain a better fit with the observed data. Although constructed in a very thoughtful manner, Chessmetrics is not a statistically well-founded method and is a filtering algorithm that disregards information from future games.

The first approach to the historical rating problem with a solid statistical foundation was developed by Mark Glickman, chairman of the USCF Rating Committee. Glicko 1 & 2 [5] are Bayesian rating systems that address a number of drawbacks of the Elo system while still being based on the Bradley-Terry paired-comparison method [1] used by modern Elo. Glickman models skills as Gaussian variables whose variances indicate the reliability of the skill estimate, an idea later adopted in the TrueSkill model as well. Glicko 2 adds volatility measures, indicating the degree of expected fluctuation in a player's rating. After an initial estimate past estimations are smoothed by propagating information back in time.

The second statistically well founded approach are Rod Edwards's Edo Historical Chess Ratings [2], which are also based on the *Bradley-Terry* model but have been applied only to historical games from the 19th century. In order to model skill dynamics Edwards considers

the same player at different times as several distinct players, whose skills are linked together by a set of virtual games which are assumed to end in draws. While Edo incorporates a dynamics model via virtual games and returns uncertainty measures in terms of the estimator's variance it is not a full Bayesian model and provides neither posterior distributions over skills, nor does it explicitly model draws.

In light of the above previous work on historical chess ratings the goal of this paper is to introduce a fully probabilistic model of chess ratings through time which explicitly accounts for draws and provides posterior distributions of skills that reflect the reliability of the estimate at every point in time.

## 3   Models for Ranking through Time

This paper strongly builds on the original TrueSkill paper [6]. Although TrueSkill is applicable to the case of multiple team games, we will only consider the two player case for this application to chess. It should be clear, however, that the methods presented can equally well be used for games with any number of teams competing.

Consider a game such as chess in which a number of, say, $N$ players $\{1, \ldots, N\}$ are competing over a period of $T$ time steps, say, years. Denote the series of game outcomes between two players $i$ and $j$ in year $t$ by $\mathbf{y}_{ij}^t(k) \in \{+1, -1, 0\}$ where $k \in \{1, \ldots, K_{ij}^t\}$ denotes the number of game outcomes available for that pair of players in that year. Furthermore, let $y = +1$ if player $i$ wins, $y = -1$ if player $j$ wins and $y = 0$ in case of a draw.

### 3.1   Vanilla TrueSkill

In the Vanilla TrueSkill system, each player $i$ is assumed to have an unknown skill $s_i^t \in \mathbb{R}$ at time $t$. We assume that a game outcome $y_{ij}^t(k)$ is generated as follows. For each of the two players $i$ and $j$ performances $p_{ij}^t(k)$ and $p_{ji}^t(k)$ are drawn according to $p\left(p_{ij}^t(k) \mid s_i^t\right) = \mathcal{N}\left(p_{ij}^t(k); s_i^t, \beta^2\right)$. The outcome $y_{ij}^t(k)$ of the game between players $i$ and $j$ is then determined as

$$y_{ij}^t(k) := \begin{cases} +1 & \text{if} & p_{ij}^t(k) > p_{ji}^t(k) + \varepsilon \\ -1 & \text{if} & p_{ij}^t(k) > p_{ij}^t(k) + \varepsilon \\ 0 & \text{if} & \left|p_{ij}^t(k) - p_{ji}^t(k)\right| \le \varepsilon \end{cases} ,$$

where the parameter $\varepsilon > 0$ is the draw margin. In order to infer the unknown skills $s_i^t$ the TrueSkill model assumes a factorising Gaussian prior $p\left(s_i^0\right) = \mathcal{N}\left(s_i^0; \mu_0, \sigma_0^2\right)$ over skills and a Gaussian drift of skills between time steps given by $p\left(s_i^t \mid s_i^{t-1}\right) = \mathcal{N}\left(s^t; s^{t-1}, \tau^2\right)$. The model can be well described as a factor graph (see Figure 1, left) which clarifies the factorisation assumptions of the model and allows to develop efficient (approximate) inference algorithms based on message passing (for details see [6])

In the Vanilla TrueSkill algorithm denoting the winning player by $W$ and the losing player by $L$ and dropping the time index for now, approximate Bayesian inference (Gaussian density filtering [7]) leads to the following update equations for $\mu_W$, $\mu_L$, $\sigma_W$ and $\sigma_L$.

$$\mu_W \leftarrow \mu_W + \frac{\sigma_W^2}{c_{ij}} \cdot v\left(\frac{\mu_W - \mu_L}{c_{ij}}, \frac{\varepsilon}{c_{ij}}\right) \quad \text{and} \quad \sigma_W \leftarrow \sigma_W \sqrt{1 - \frac{\sigma_W^2}{c_{ij}^2} \cdot w\left(\frac{\mu_W - \mu_L}{c_{ij}}, \frac{\varepsilon}{c_{ij}}\right)}$$

$$\mu_L \leftarrow \mu_L - \frac{\sigma_L^2}{c_{ij}} \cdot v\left(\frac{\mu_W - \mu_L}{c_{ij}}, \frac{\varepsilon}{c_{ij}}\right) \quad \text{and} \quad \sigma_L \leftarrow \sigma_L \sqrt{1 - \frac{\sigma_L^2}{c_{ij}^2} \cdot w\left(\frac{\mu_W - \mu_L}{c_{ij}}, \frac{\varepsilon}{c_{ij}}\right)} .$$

The overall variance is $c_{ij}^2 = 2\beta^2 + \sigma_W^2 + \sigma_L^2$ and the two functions $v$ and $w$ are given by

$$v(t, \alpha) := \frac{\mathcal{N}(t - \alpha; 0, 1)}{\Phi(t - \alpha)} \quad \text{and} \quad w(t, \alpha) := v(t, \alpha) \cdot (v(t, \alpha) + (t - \alpha)) .$$

For the case of a draw we have the following update equations:

$$\mu_i \leftarrow \mu_i + \frac{\sigma_i^2}{c_{ij}} \cdot \tilde{v}\left(\frac{\mu_i - \mu_i}{c_{ij}}, \frac{\varepsilon}{c_{ij}}\right) \quad \text{and} \quad \sigma_i \leftarrow \sigma_i \sqrt{1 - \frac{\sigma_i^2}{c_{ij}^2} \cdot \tilde{w}\left(\frac{\mu_i - \mu_i}{c_{ij}}, \frac{\varepsilon}{c_{ij}}\right)},$$

and similarly for player $j$. Defining $d := \alpha - t$ and $s := \alpha + t$ then $\tilde{v}$ and $\tilde{w}$ are given by

$$\tilde{v}(t, \alpha) := \frac{\mathcal{N}(-s; 0, 1) - \mathcal{N}(d; 0, 1)}{\Phi(d) - \Phi(-s)} \quad \text{and} \quad \tilde{w}(t, \alpha) := \tilde{v}^2(t, \alpha) + \frac{(d)\,\mathcal{N}(d; 0, 1) - (s)\,\mathcal{N}(s; 0, 1)}{\Phi(d) - \Phi(-s)}.$$

In order to approximate the skill parameters $\mu_i^t$ and $\sigma_i^t$ for all players $i \in \{1, \ldots, N\}$ at all times $t \in \{0, \ldots, T\}$ the Vanilla TrueSkill algorithm initialises each skill belief with $\mu_i^0 \leftarrow \mu_0$ and $\sigma_i^0 \leftarrow \sigma_0$. It then proceeds through the years $t \in \{1 \ldots T\}$ in order, goes through the game outcomes $y_{ij}^t(k)$ in random order and updates the skill beliefs according to the equations above.

## 3.2 TrueSkill through Time (TTT)

The Vanilla TrueSkill algorithm suffers from two major disadvantages:

1. Inference within a given year $t$ depends on the random order chosen for the updates. Since no knowledge is assumed about game outcomes within a given year, the results of inference should be independent of the order of games within a year.

2. Information across years is only propagated forward in time. More concretely, if player A beats player B and player B later turns out to be very strong (i.e., as evidenced by him beating very strong player C repeatedly), then Vanilla TrueSkill cannot propagate that information backwards in time to correct player A's skill estimate upwards.

Both problems can be addressed by extending the Gaussian density filtering to running full expectation propagation (EP) until convergence [7]. The basic idea is to update repeatedly on the same game outcomes but making sure that the effect of the previous update on that game outcome is removed before the new effect is added. This way, the model remains the same but the inferences are less approximate.

More specifically, we go through the game outcomes $\mathbf{y}_{ij}^t$ within a year $t$ several times until convergence. The update for a game outcome $y_{ij}^t(k)$ is performed in the same way as before but saving the upward messages $m_{f\left(p_{ij}^t(k), s_i^t\right) \to s_i^t}\left(s_i^t\right)$ which describe the effect of the updated performance $p_{ij}^t(k)$ on the underlying skill $s_i^t$. When game outcome $y_{ij}^t(k)$ comes up for update again, the new downward message $m_{f\left(p_{ij}^t(k), s_i^t\right) \to p_{ij}^t(k)}\left(p_{ij}^t(k)\right)$ can be calculated by

$$m_{f\left(p_{ij}^t(k), s_i^t\right) \to p_{ij}^t(k)}\left(p_{ij}^t(k)\right) = \int_{-\infty}^{\infty} f\left(p_{ij}^t(k), s_i^t\right) \frac{p\left(s_i^t\right)}{m_{f\left(p_{ij}^t(k), s_i^t\right) \to s_i^t}\left(s_i^t\right)} ds_i^t,$$

thus effectively dividing out the earlier upward message to avoid double counting. The integral above is easily evaluated since the messages as well as the marginals $p\left(s_i^t\right)$ have been assumed Gaussian. The new downward message serves as the effective prior belief on the performance $p_i^t(k)$. At convergence, the dependency of the inferred skills on the order of game outcomes vanishes.

The second problem is addressed by performing inference for TrueSkill through time (TTT), i.e. by repeatedly smoothing forward and backward in time. The first forward pass of TTT is identical to the inference pass of Vanilla TrueSkill except that the forward messages $m_{f\left(s_i^{t-1}, s_i^t\right) \to s_i^t}\left(s_i^t\right)$ are stored. They represent the influence of skill estimate $s_i^{t-1}$ at time $t-1$ on skill estimate $s_i^t$ at time $t$. In the backward pass, these messages are then used to calculate the new backward messages $m_{f\left(s_i^{t-1}, s_i^t\right) \to s_i^{t-1}}\left(s_i^{t-1}\right)$, which effectively serve as the new prior for time step $t-1$,

$$m_{f\left(s_i^{t-1}, s_i^t\right) \to s_i^{t-1}}\left(s_i^{t-1}\right) = \int_{-\infty}^{\infty} f\left(s_i^{t-1}, s_i^t\right) \frac{p\left(s_i^t\right)}{m_{f\left(s_i^{t-1}, s_i^t\right) \to s_i^t}\left(s_i^t\right)} ds_i^t.$$

This procedure is repeated forward and backward along the time series of skills until convergence. The backward passes make it possible to propagate information from the future into the past.

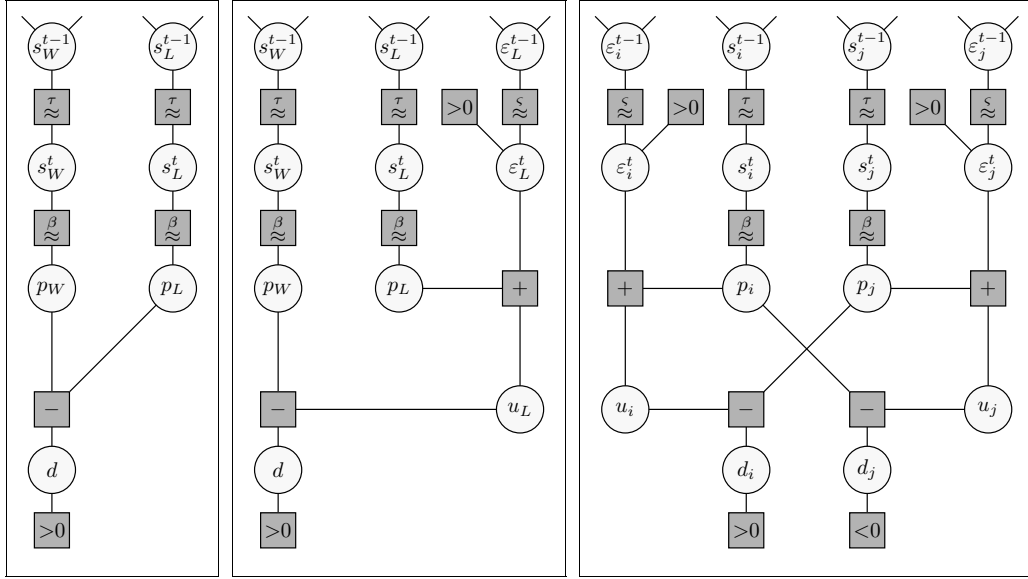

Figure 1: Factor graphs of single game outcomes for TTT (left) and TTT-D. In the left graph there are three types of variables: skills $s$, performances $p$, performance differences $d$. In the TTT-D graphs there are two additional types: draw margins $\varepsilon$ and winning thresholds $u$.

The graphs only require three different types of factors: factor $\boxed{\overset{\tau}{\approx}}$ takes the form $\mathcal{N}\left(\cdot;\cdot,\tau^2\right)$, factor $\boxed{>0}$ takes the form $\mathbb{I}\left(\cdot>0\right)$ and factor $\boxed{\pm}$ takes the form $\mathbb{I}\left(\cdot\pm\cdot=\cdot\right)$.

## 3.3 TTT with Individual Draw Margins (TTT-D)

From exploring the data it is known that the probability of draw not only increases markedly through the history of chess, but is also positively correlated with playing skill and even varies considerably across individual players. We would thus like to extend the TrueSkill model to incorporate another player-specific parameter which indicates a player's ability to force a draw. Suppose each player $i$ at every time-step $t$ is characterised by an unknown skill $s_i^t \in \mathbb{R}$ and a player-specific draw margin $\varepsilon_i^t > 0$. Again, performances $p_{ij}^t(k)$ and $p_{ji}^t(k)$ are drawn according to $p\left(p_{ij}^t(k)\,|s_i^t\right) = \mathcal{N}\left(p_{ij}^t(k)\,;s_i^t,\beta^2\right)$. In this model a game outcome $y_{ij}^t(k)$ between players $i$ and $j$ at time $t$ is generated as follows:

$$
y_{ij}^t(k) = \begin{cases}
+1 & \text{if} & p_{ij}^t(k) > p_{ji}^t(k) + \varepsilon_j^t \\
-1 & \text{if} & p_{ji}^t(k) > p_{ij}^t(k) + \varepsilon_i^t \\
0 & \text{if} & -\varepsilon_i^t \le p_{ij}^t(k) - p_{ji}^t(k) \le \varepsilon_j^t
\end{cases} .
$$

In addition to the Gaussian assumption about player skills as in the Vanilla TrueSkill model of Section 3.1 we assume a factorising Gaussian distribution for the player-specific draw margins $p\left(\varepsilon_i^0\right) = \mathcal{N}\left(\varepsilon_i^0;\nu_0,\varsigma_0^2\right)$ and a Gaussian drift of draw margins between time steps given by $p\left(\varepsilon_i^t|\varepsilon_i^{t-1}\right) = \mathcal{N}\left(\varepsilon^t;\varepsilon^{t-1},\varsigma^2\right)$. The factor graph for the case of win/loss is shown in Figure 1 (centre) and for the case of a draw in Figure 1 (right). Note, that the positivity of the player-specific draw margins at each time step $t$ is enforced by a factor $\boxed{>0}$.

Inference in the TTT-D model is again performed by expectation propagation, both within a given year $t$ as well as across years in a forward backward manner. Note that in this model the current belief about the skill of a player is represented by four numbers: $\mu_i^t$ and $\sigma_i^t$ for the skill and $\nu_i^t$ and $\varsigma_i^t$ for the player-specific draw margin. Players with a high value of $\nu_i^t$ can be thought of as having the ability to achieve a draw against strong players, while players with a high value of $\mu_i^t$ have the ability to achieve a win.

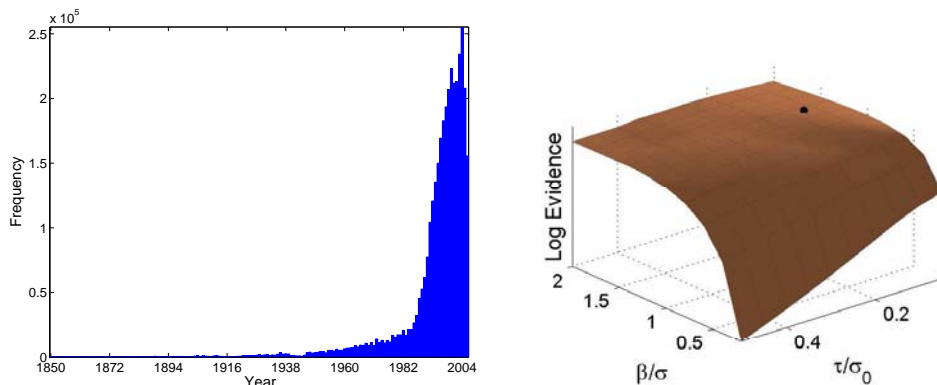

Figure 2: **(Left)** Distribution over number of recorded match outcomes played per year in the ChessBase database. **(Right)** The log-evidence $P(\mathbf{y}|\beta,\tau)$ for the TTT model as a function of the variation of player performance, $\beta$, and skill dynamics, $\tau$. The maximizing parameter settings are indicated by a black dot.

## 4 Experiments and Results

Our experiments are based on a data-set of chess match outcomes collected by ChessBase[1]. This database is the largest top-class annotated database in the world and covers more than 3.5 million chess games from 1560 to 2006 played between $\approx$200,000 unique players. From this database, we selected all the matches between 1850 (the birth of modern Chess) and 2006. This results in 3,505,366 games between 206,059 unique players. Note that a large proportion of games was collected between 1987 and 2006 (see Figure 2 (left)).

Our implementation of the TrueSkill through Time algorithms was done in F#[2] and builds a factor graph with approximately 11,700,000 variables and 15,200,000 factors (TTT) or 18,500,000 variables and 27,600,000 factors (TTT-D). The whole schedule allocates no more than 6 GB (TTT) or 11 GB (TTT-D) and converges in less than 10 minutes (TTT)/20 minutes (TTT-D) of CPU time on a standard Pentium 4 machine. The code for this analysis will be made publicly available.

In the first experiment, we built the TTT model for the above mentioned collection of Chess games. The draw margin was chosen such that the a-priori probability of draw between two equally skilled players matches the overall draw probability of 30.3%. Moreover, the model has a translational invariance in the skills and a scale invariance in $\beta/\sigma_0$ and $\tau/\sigma_0$. Thus, we fixed $\mu_0 = 1200$, $\sigma_0 = 400$ and computed the log-evidence $L := P(\mathbf{y}|\beta,\tau)$ for varying values of $\beta$ and $\tau$ (see Figure 2 (right)). The plots show that the model is very robust to setting these two parameters except if $\beta$ is chosen too small. Interestingly, the log-evidence is neither largest for $\tau \gg 0$ (complete de-coupling) nor for $\tau \to 0$ (constant skill over life-time) indicating that it is important to model the dynamics of Chess players. Note that the log-evidence is $L_{\mathrm{TTT}} = -3,953,997$, larger than that of the naive model ($L_{\mathrm{naive}} = -4,228,005$) which always predicts 30.3% for a draw and correspondingly for win/loss[3]. In a second experiment, we picked the optimal values $(\beta^*, \tau^*) = (480, 60)$ for TTT and optimised the remaining prior and dynamics parameters of TTT-D to arrive at a model with a log-evidence of $L_{\mathrm{TTT-D}} = -3,661,813$.

In Figure 3 we have plotted the skill evolution for some well–known players of the last 150 years when fitting the TTT model ($\mu^t, \sigma^t$ are shown). In Figure 4 the skill evolution of the same players is plotted when fitting the TTT-D model; the dashed lines show $\mu^t + \varepsilon^t$

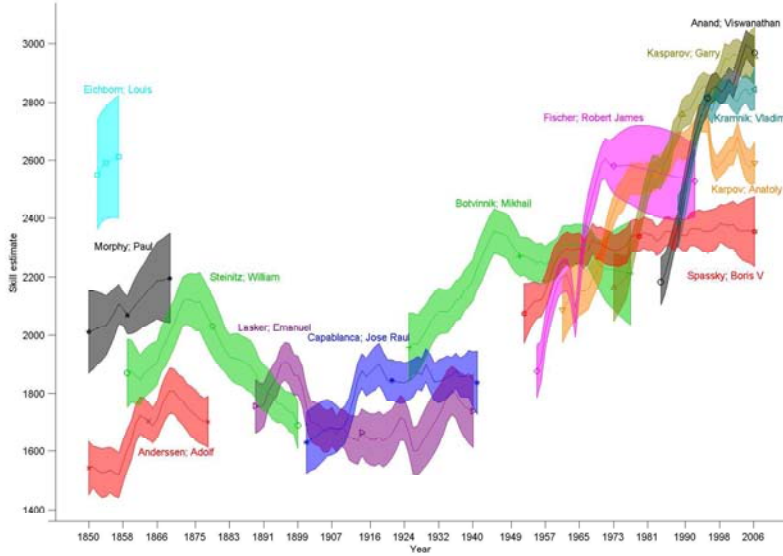

Figure 3: Skill evolution of top Chess players with TTT; see text for details.

whereas the solid lines display $\mu^t$; for comparisons we added the $\mu^t$ of the TTT model as dotted lines.

As a first observation, the uncertainties always grow towards the beginning and end of a career since they are not constrained by past/future years. In fact, for *Bobby Fischer* the uncertainty grows very large in his 20 years of inactivity (1972–1992). Moreover, there seems to be a noticeable increase in overall skill since the 1980's. Looking at Figure 4 we see that players have different abilities to force a draw; the strongest player to do so is *Boris Spassky* (1937–). This ability got stronger after 1975 which explains why the model with a fixed draw margin estimates Spassky's skill larger.

Looking at individual players we see that *Paul Morphy* (1837–1884), "The Pride and Sorrow of Chess", is particularly strong when comparing his skill to those of his contemporaries in the next 80 years. He is considered to have been the greatest chess master of his time, and this is well supported by our analysis. *"Bobby" Fischer* (1943–) tied with *Boris Spassky* at the age of 17 and later defeated Spassky in the "Match of the Century" in 1972. Again, this is well supported by our model. Note how the uncertainty grows during the 20 years of inactivity (1972–1992) but starts to shrink again in light of the (future) re-match of Spassky and Fischer in 1992 (which Fischer won). Also, Fischer is the only one of these players whose $\varepsilon^t$ *decreased* over time—when he was active, he was known for the large margin by which he won!

Finally, *Garry Kasparov* (1963–) is considered the strongest Chess player of all time. This is well supported by our analysis. In fact, based on our analysis Kasparov is still considerably stronger than *Vladimir Kramnik* (1975–) but a contender for the crown of strongest player in the world is *Viswanathan Anand* (1969–), a former FIDE world champion.

## 5   Conclusion

We have extended the Bayesian rating system TrueSkill to provide player ratings through time on a unified scale. In addition, we introduced a new model that tracks player-specific draw margins and thus models the game outcomes even more precisely. The resulting factor graph model for our large ChessBase database of game outcomes has 18.5 million nodes and 27.6 million factors, thus constituting one of the largest non-trivial Bayesian models ever

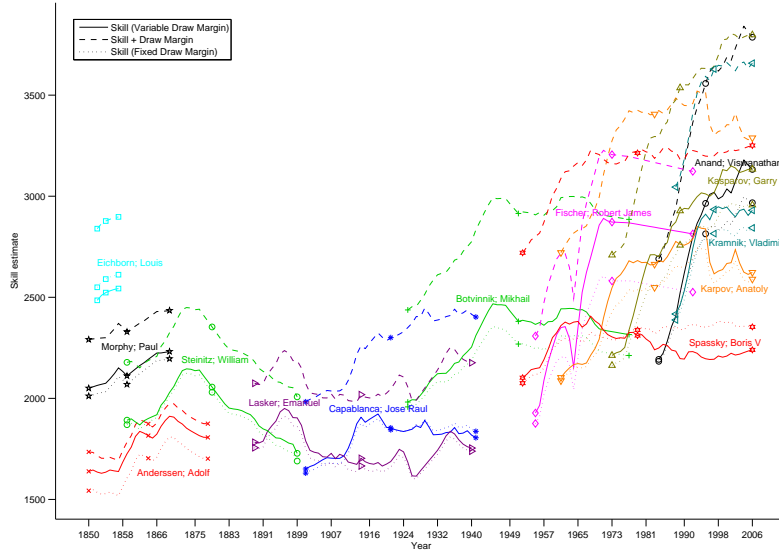

Figure 4: Skill evolution of top Chess players with TTT-D; see text for details.

tackled. Full approximate inference takes a mere 20 minutes in our F# implementation and thus demonstrates the efficiency of EP in appropriately structured factor graphs.

One of the key questions provoked by this work concerns the comparability of skill estimates across different eras of chess history. Can we directly compare Fischer's rating in 1972 with Kasparov's in 1991? Edwards [2] points out that we would not be able to detect any skill improvement if two players of equal skill were to learn about a skill-improving breakthrough in chess theory at the same time but would only play against each other. However, this argument does not rule out the possibility that with more players and chess knowledge flowing less perfectly the improvement may be detectable. After all, we do see a marked improvement in the average skill of the top players.

In future work, we would like to address the issue of skill calibration across years further, e.g., by introducing a latent variable for each year that serves as the prior for new players joining the pool. Also, it would be interesting to model the effect of playing white rather than black.

## Footnotes

[1]For more information, see *http://www.bcmchess.co.uk/softdatafrcb.html*.

[2]For more details, see *http://research.microsoft.com/fsharp/fsharp.aspx*.

[3]Leakage due to approximate inference.

# References

[1] H. A. David. *The method of paired comparisons.* Oxford University Press, New York, 1988.

[2] R. Edwards. Edo historical chess ratings. http://members.shaw.ca/edo1/.

[3] A. E. Elo. *The rating of chess players: Past and present.* Arco Publishing, New York, 1978.

[4] M. E. Glickman. A comprehensive guide to chess ratings. *Amer. Chess Journal*, 3:59–102, 1995.

[5] M. E. Glickman. Parameter estimation in large dynamic paired comparison experiments. *Applied Statistics*, 48:377–394, 1999.

[6] R. Herbrich, T. Minka, and T. Graepel. TrueSkill(TM): A Bayesian skill rating system. In *Advances in Neural Information Processing Systems 20*, 2007.

[7] T. Minka. *A family of algorithms for approximate Bayesian inference.* PhD thesis, MIT, 2001.

[8] J. Sonas. Chessmetrics. http://db.chessmetrics.com/.

